# Mean field methods for classification with Gaussian processes

**Manfred Opper**
Neural Computing Research Group
Division of Electronic Engineering and Computer Science
Aston University Birmingham B4 7ET, UK.
opperm@aston.ac.uk

**Ole Winther**
Theoretical Physics II, Lund University, Sölvegatan 14 A
S-223 62 Lund, Sweden
CONNECT, The Niels Bohr Institute, University of Copenhagen
Blegdamsvej 17, 2100 Copenhagen Ø, Denmark
winther@thep.lu.se

## Abstract

We discuss the application of TAP mean field methods known from the Statistical Mechanics of disordered systems to Bayesian classification models with Gaussian processes. In contrast to previous approaches, no knowledge about the distribution of inputs is needed. Simulation results for the Sonar data set are given.

## 1  Modeling with Gaussian Processes

Bayesian models which are based on Gaussian prior distributions on function spaces are promising non-parametric statistical tools. They have been recently introduced into the Neural Computation community (Neal 1996, Williams & Rasmussen 1996, Mackay 1997). To give their basic definition, we assume that the likelihood of the output or target variable $\tau$ for a given input $\mathbf{s} \in R^N$ can be written in the form $p(\tau|h(\mathbf{s}))$ where $h : R^N \to R$ is a priori assumed to be a Gaussian random field. If we assume fields with zero prior mean, the statistics of $h$ is entirely defined by the second order correlations $C(\mathbf{s}, \mathbf{s}') \doteq E[h(\mathbf{s})h(\mathbf{s}')]$, where $E$ denotes expectations

with respect to the prior. Interesting examples are

$$C(\mathbf{s}, \mathbf{s}') = \frac{2}{\pi} \arcsin \left( \frac{\sum_i w_i s_i s_i'}{\sqrt{(1 + \sum_i w_i s_i s_i)(1 + \sum_i w_i s_i' s_i')}} \right) \quad (1)$$

$$C(\mathbf{s}, \mathbf{s}') = \exp \left( -\frac{1}{2} \sum_i w_i (s_i - s_i')^2 \right) \quad (2)$$

The choice (1) can be motivated as a limit of a two-layered neural network with infinitely many hidden units with factorizable input-hidden weight priors (Williams 1997). $w_i$ are hyperparameters determining the relevant prior lengthscales of $h(\mathbf{s})$. The simplest choice $C(\mathbf{s}, \mathbf{s}') = \sum_i w_i s_i s_i'$ corresponds to a single layer perceptron with independent Gaussian weight priors.

In this Bayesian framework, one can make predictions on a novel input $\mathbf{s}$ after having received a set $D_m$ of $m$ training examples $(\tau^\mu, \mathbf{s}^\mu)$, $\mu = 1, \ldots, m$ by using the posterior distribution of the field at the test point $\mathbf{s}$ which is given by

$$p(h(\mathbf{s})|D_m) = \int p(h(\mathbf{s})|\{h^\nu\}) \, p(\{h^\nu\}|D_m) \prod_\mu dh^\mu. \quad (3)$$

$p(h(\mathbf{s})|\{h^\nu\})$ is a conditional Gaussian distribution and

$$p(\{h^\nu\}|D_m) = \frac{1}{Z} p(\{h^\nu\}) \prod_\mu p(\tau^\mu|h^\mu). \quad (4)$$

is the posterior distribution of the field variables at the training points. $Z$ is a normalizing partition function and

$$p(\{h^\mu\}) = \frac{1}{\sqrt{(2\pi)^m \det C}} e^{-\frac{1}{2} \sum_{\mu\nu} h^\mu (C^{-1})_{\mu\nu} h^\nu}. \quad (5)$$

is the prior distribution of the fields at the training points. Here, we have introduced the abbreviations $h^\mu = h(\mathbf{s}^\mu)$ and $C_{\mu\nu} \doteq C(\mathbf{s}^\mu, \mathbf{s}^\nu)$.

The major technical problem of this approach comes from the difficulty in performing the high dimensional integrations. Non-Gaussian likelihoods can be only treated by approximations, where e.g. Monte Carlo sampling (Neal 1997), Laplace integration (Barber & Williams 1997) or bounds on the likelihood (Gibbs & Mackay 1997) have been used so far. In this paper, we introduce a further approach, which is based on a mean field method known in the Statistical Physics of disordered systems (Mézard, Parisi & Virasoro 1987).

We specialize on the case of a binary classification problem, where a binary class label $\tau = \pm 1$ must be predicted using a training set corrupted by i.i.d label noise. The likelihood for this problem is taken as

$$p(\tau|h) = \kappa + (1 - 2\kappa)\Theta(\tau h),$$

where $\kappa$ is the probability that the true classification label is corrupted, i.e. flipped and the step function, $\Theta(x)$ is defined as $\Theta(x) = 1$ for $x > 0$ and 0 otherwise. For such a case, we expect that (by the non-smoothness of the model), e.g. Laplace's method and the bounds introduced in (Gibbs & Mackay 1997) are not directly applicable.

## 2 Exact posterior averages

In order to make a prediction on an input $\mathbf{s}$, ideally the label with maximum posterior probability should be chosen, i.e. $\tau^{\text{Bayes}} = \text{argmax}_\tau\, p(\tau|D_m)$, where the predictive probability is given by $p(\tau|D_m) = \int dh\, p(\tau|h)\, p(h|D_m)$. For the binary case the Bayes classifier becomes $\tau^{\text{Bayes}} = \text{sign}\langle \text{sign} h(\mathbf{s})\rangle$, where we throughout the paper let brackets $\langle \ldots \rangle$ denote posterior averages. Here, we use a somewhat simpler approach by using the prediction

$$\tau = \text{sign}(\langle h(\mathbf{s})\rangle) \ .$$

This would reduce to the ideal prediction, when the posterior distribution of $h(\mathbf{s})$ is symmetric around its mean $\langle h(\mathbf{s})\rangle$. The goal of our mean field approach will be to provide a set of equations for approximately determining $\langle h(\mathbf{s})\rangle$. The starting point of our analysis is the partition function

$$Z = \int \prod_\mu \frac{dx^\mu dh^\mu}{2\pi i} \prod_\mu p(\tau^\mu|h^\mu) e^{\frac{1}{2}\sum_{\mu,\nu} C_{\mu\nu} x^\mu x^\nu - \sum_\mu h^\mu x^\mu} \ , \tag{6}$$

where the new auxiliary variables $x^\mu$ (integrated along the imaginary axis) have been introduced in order to get rid of $C^{-1}$ in (5).

It is not hard to show from (6) that the posterior averages of the fields at the $m$ training inputs and at a new test point $\mathbf{s}$ are given by

$$\langle h^\mu \rangle = \sum_\nu C_{\mu\nu}\langle x^\nu \rangle \qquad \langle h(\mathbf{s})\rangle = \sum_\nu C(\mathbf{s}, \mathbf{s}^\nu)\langle x^\nu\rangle. \tag{7}$$

We have thus reduced our problem to the calculation of the "microscopic orderparameters" $\langle x^\mu \rangle$.[1] Averages in Statistical Physics can be calculated from derivatives of $-\ln Z$ with respect to small external fields, which are then set to zero. An equivalent formulation uses the *Legendre transform* of $-\ln Z$ as a function of the expectations, which in our case is given by

$$G(\{\langle x^\mu \rangle, \langle (x^\mu)^2\rangle\}) = -\ln Z(\gamma^\mu, \lambda) + \sum_\mu \langle x^\mu\rangle\gamma^\mu + \frac{1}{2}\sum_\mu \lambda_\mu \langle (x^\mu)^2\rangle \ . \tag{8}$$

with

$$Z(\{\gamma^\mu, \lambda_\mu\}) = \int \prod_\mu \frac{dx^\mu dh^\mu}{2\pi i} \prod_\mu p(\tau^\mu|h^\mu) e^{\frac{1}{2}\sum_{\mu,\nu}(\lambda_\mu \delta_{\mu\nu} + C_{\mu\nu})x^\mu x^\nu + \sum_\mu x^\mu(\gamma^\mu - h^\mu)} \ . \tag{9}$$

The additional averages $\langle (x^\mu)^2\rangle$ have been introduced, because the dynamical variables $x^\mu$ (unlike Ising spins) do not have fixed length. The external fields $\gamma^\mu, \lambda_\mu$ must be eliminated from $\frac{\partial G}{\partial \lambda_\mu} = \frac{\partial G}{\partial \gamma^\mu} = 0$ and the true expectation values of $x^\mu$ and $(x^\mu)^2$ are those which satisfy $\frac{\partial G}{\partial \langle (x^\mu)^2\rangle} = \frac{\partial G}{\partial \langle x^\mu\rangle} = 0$.

## 3 Naive mean field theory

So far, this description does not give anything new. Usually $G$ cannot be calculated exactly for the non-Gaussian likelihood models of interest. Nevertheless, based on mean field theory (MFT) it is possible to guess an approximate form for $G$.

Mean field methods have found interesting applications in Neural Computing within the framework of *ensemble learning*, where the the exact posterior distribution is approximated by a simpler one using product distributions in a variational treatment. Such a "standard" mean field method for the posterior of the $h^\mu$ (for the case of Gaussian process classification) is in preparation and will be discussed somewhere else. In this paper, we suggest a different route, which introduces nontrivial corrections to a simple or "naive" MFT for the variables $x^\mu$. Besides the variational method (which would be purely formal because the distribution of the $x^\mu$ is complex and does not define a probability), there are other ways to define the simple MFT. E.g., by truncating a perturbation expansion with respect to the "interactions" $C_{\mu\nu}$ in $G$ after the first order (Plefka 1982). These approaches yield the result

$$G \approx G_{naive} = G_0 - \frac{1}{2} \sum_\mu C_{\mu\mu} \langle (x^\mu)^2 \rangle - \frac{1}{2} \sum_{\mu,\nu,\mu\neq\mu} C_{\mu\nu} \langle x^\mu \rangle \langle x^\nu \rangle \ . \tag{10}$$

$G_0$ is the contribution to $G$ for a model without any interactions i.e. when $C_{\mu\nu} = 0$ in (9), i.e. it is the Legendre transform of

$$- \ln Z_0 = \sum_\mu \ln \left[ \kappa + (1 - 2\kappa)\Phi \left( \tau^\mu \frac{\gamma^\mu}{\sqrt{\lambda^\mu}} \right) \right] \ ,$$

where $\Phi(z) = \int_{-\infty}^z \frac{dt}{\sqrt{2\pi}} e^{-t^2/2}$ is an error function. For simple models in Statistical Physics, where *all* interactions $C_{\mu\nu}$ are positive and equal, it is easy to show that $G_{naive}$ will become exact in the limit of an infinite number of variables $x^\mu$. Hence, for systems with a large number of nonzero interactions having the same orders of magnitude, one may expect that the approximation is not too bad.

## 4   The TAP approach

Nevertheless, when the interactions $C_{\mu\nu}$ can be both positive and negative (as one would expect e.g. when inputs have zero mean), even in the thermodynamic limit and for nice distributions of inputs, an additional contribution $\Delta G$ must be added to the "naive" mean field theory (10). Such a correction (often called an *Onsager reaction term*) has been introduced for a spin glass model by (Thouless, Anderson & Palmer 1977) (TAP). It was later applied to the statistical mechanics of single layer perceptrons by (Mézard 1989) and then generalized to the Bayesian framework by (Opper & Winther 1996, 1997). For an application to multilayer networks, see (Wong 1995). In the thermodynamic limit of infinitely large dimension of the input space, and for nice input distributions, the results can be shown coincide with the results of the replica framework. The drawback of the previous derivations of the TAP MFT for neural networks was the fact that special assumptions on the input distribution had been made and certain fluctuating terms have been replaced by their averages over the distribution of random data, which in practice would not be available. In this paper, we will use the approach of (Parisi & Potters 1995), which allows to circumvent this problem. They concluded (applied to the case of a spin model with random interactions of a specific type), that the functional form of $\Delta G$ should not depend on the type of the "single particle" contribution $G_0$. Hence, one may use any model in $G_0$, for which $G$ can be calculated exactly (e.g. the Gaussian regression model) and subtract the naive mean field contribution to obtain the

desired $\Delta G$. For the sake of simplicity, we have chosen the even simpler model $p(\tau^\mu|h^\mu) \sim \delta(h^\mu)$ without changing the final result. A lengthy but straightforward calculation for this problem leads to the result

$$\Delta G = \frac{1}{2}\ln\det(\Lambda + C) + \frac{1}{2}\sum_\mu (\Lambda_\mu - C_{\mu\mu})R_\mu + \frac{m}{2} + \frac{1}{2}\sum_\mu \ln(-R_\mu) \ . \tag{11}$$

with $R_\mu \doteq \langle(x^\mu)^2\rangle - \langle x^\mu\rangle^2$. The $\Lambda_\mu$ must be eliminated using $\frac{\partial G}{\partial \Lambda_\mu} = 0$, which leads to the equation

$$R_\mu = -\left[(\Lambda + C)^{-1}\right]_{\mu\mu} . \tag{12}$$

Note, that with this choice, the TAP mean field theory becomes exact for Gaussian likelihoods, i.e. for standard regression problems.

Finally, setting the derivatives of $G_{TAP} = G_{naive} + \Delta G$ with respect to the 4 variables $\langle x^\mu\rangle, \langle(x^\mu)^2\rangle, \gamma_\mu, \lambda_\mu$ equal to zero, we obtain the equations

$$\gamma^\mu = \sum_\nu C_{\mu\nu}\langle x^\nu\rangle - \lambda_\mu\langle x^\mu\rangle \qquad \lambda_\mu = -(\Lambda_\mu + \frac{1}{R_\mu}) \tag{13}$$

$$\langle x^\mu\rangle = \frac{\tau^\mu}{\sqrt{\lambda_\mu}}\frac{(1-2\kappa)D\left(\frac{\gamma^\mu}{\sqrt{\lambda_\mu}}\right)}{\kappa + (1-2\kappa)\Phi\left(\tau^\mu\frac{\gamma^\mu}{\sqrt{\lambda_\mu}}\right)} \qquad R_\mu = -\langle x^\mu\rangle\left(\frac{\gamma^\mu}{\lambda_\mu} + \langle x^\mu\rangle\right) \ ,$$

where $D(z) = e^{-z^2/2}/\sqrt{2\pi}$ is the Gaussian measure. These eqs. have to be solved numerically together with (12). In contrast, for the naive MFT, the simpler result $\lambda_\mu = C_{\mu\mu}$ is found.

## 5 Simulations

Solving the nonlinear system of equations (12,13) by iteration turns out to be quite straightforward. For some data sets to get convergence, one has to add a diagonal term $v$ to the covariance matrix $C$: $C_{ij} \to C_{ij} + \delta_{ij}v$. It may be shown that this term corresponds to learning with Gaussian noise (with variance $v$) added the Gaussian random field.

Here, we present simulation results for a single data set, the *Sonar – Mines versus Rocks* using the same training/test set split as in the original study by (Gorman & Sejnowski 1988). The input data were pre-processed by linear rescaling such that over the training set each input variable has zero mean and unit variance. In some cases the mean field equations failed to converge using the raw data.

A further important feature of TAP MFT is the fact that the method also gives an approximate leave-one-out estimator for the generalization error, $\epsilon_{loo}$ expressed in terms of the solution to the mean field equations (see (Opper & Winther 1996, 1997) for more details). It is also possible to derive a leave-one-out estimator for the naive MFT (Opper & Winther to be published).

Since we so far haven't dealt with the problem of automatically estimating the hyperparameters, their number was drastically reduced by setting $w_i = \frac{1}{\sigma^2 N}$ in the covariances (1) and (2). The remaining hyperparameters, $\sigma^2$, $\kappa$ and $v$ were chosen

Table 1: The result for the Sonar data.

| Algorithm | Covariance Function | $\epsilon_{\text{test}}$ | $\epsilon_{\text{loo}}^{\text{exact}}$ | $\epsilon_{\text{loo}}$ |
|---|---|---|---|---|
| TAP Mean Field | (1) | 0.183 | 0.260 | 0.260 |
|  | (2) | 0.077 | 0.212 | 0.212 |
| Naive Mean Field | (1) | 0.154 | 0.269 | 0.269 |
|  | (2) | 0.077 | 0.221 | 0.221 |
| Back-Prop | Simple Perceptron | 0.269($\pm$0.048) |  |  |
|  | Best 2layer – 12 Hidden | 0.096($\pm$0.018) |  |  |

as to minimize $\epsilon_{\text{loo}}$. It turned out that the lowest $\epsilon_{\text{loo}}$ was found from modeling without noise: $\kappa = v = 0$.

The simulation results are shown in table 1. The comparisons for back-propagation is taken from (Gorman & Sejnowski 1988). The solution found by the algorithm turned out to be unique, i.e. different order presentation of the examples and different initial values for the $\langle x^\mu \rangle$ converged to the same solution.

In table 1, we have also compared the estimate given by the algorithm with the exact leave-one-out estimate $\epsilon_{\text{loo}}^{\text{exact}}$ obtained by going through the training set and keeping an example out for testing and running the mean field algorithm on the rest. The estimate and exact value are in complete agreement. Comparing with the test error we see that the training set is 'hard' and the test set is 'easy'. The small difference for test error between the naive and full mean field algorithms also indicate that the mean field scheme is quite robust with respect to choice of $\lambda_\mu$.

## 6   Discussion

More work has to be done to make the TAP approach a practical tool for Bayesian modeling. One has to find better methods for solving the equations. A conversion into a direct minimization problem for a free energy maybe helpful. To achieve this, one may probably work with the real field variables $h^\mu$ instead of the imaginary $x^\mu$. A further problem is the determination of the hyperparameters of the covariance functions. Two ways seem to be interesting here. One may use the approximate free energy $G$, which is essentially the negative logarithm of the Bayesian *evidence* to estimate the most probable values of the hyperparameters. However, an estimate on the errors made in the TAP approach would be necessary. Second, one may use the built-in leave-one-out estimate to estimate the generalization error. Again an estimate on the validity of the approximation is necessary. It will further be interesting to apply our way of deriving the TAP equations to other models (Boltzmann machines, belief nets, combinatorial optimization problems), for which standard mean field theories have been applied successfully.

### Acknowledgments

This research is supported by the Swedish Foundation for Strategic Research and by the Danish Research Councils for the Natural and Technical Sciences through the Danish Computational Neural Network Center (CONNECT).

## Footnotes

[1]Although the integrations are over the imaginary axis, these expectations come out positive. This is due to the fact that the integration "measure" is complex as well.

## References

D. Barber and C. K. I. Williams, Gaussian Processes for Bayesian Classification via Hybrid Monte Carlo, in *Neural Information Processing Systems 9*, M . C. Mozer, M. I. Jordan and T. Petsche, eds., 340-346. MIT Press (1997).

M. N. Gibbs and D. J. C. Mackay, Variational Gaussian Process Classifiers, Preprint Cambridge University (1997).

R. P. Gorman and T. J. Sejnowski, Analysis of Hidden Units in a Layered Network Trained to Classify Sonar Targets, Neural Networks **1**, 75 (1988).

D. J. C. Mackay, Gaussian Processes, A Replacement for Neural Networks, NIPS tutorial 1997, May be obtained from `http://wol.ra.phy.cam.ac.uk/pub/mackay/`.

M. Mézard, The Space of interactions in Neural Networks: Gardner's Computation with the Cavity Method, J. Phys. A **22**, 2181 (1989).

M. Mézard and G. Parisi and M. A. Virasoro, *Spin Glass Theory and Beyond*, Lecture Notes in Physics, 9, World Scientific (1987).

R. Neal, *Bayesian Learning for Neural Networks*, Lecture Notes in Statistics, Springer (1996).

R. M. Neal, Monte Carlo Implementation of Gaussian Process Models for Bayesian Regression and Classification, Technical Report CRG-TR-97-2, Dept. of Computer Science, University of Toronto (1997).

M. Opper and O. Winther, A Mean Field Approach to Bayes Learning in Feed-Forward Neural Networks, Phys. Rev. Lett. **76**, 1964 (1996).

M. Opper and O. Winther, A Mean Field Algorithm for Bayes Learning in Large Feed-Forward Neural Networks, in *Neural Information Processing Systems 9*, M. C. Mozer, M. I. Jordan and T. Petsche, eds., 225-231. MIT Press (1997).

G. Parisi and M. Potters, Mean-Field Equations for Spin Models with Orthogonal Interaction Matrices, J. Phys. A: Math. Gen. **28** 5267 (1995).

T. Plefka, Convergence Condition of the TAP Equation for the Infinite-Range Ising Spin Glass, J. Phys. A **15**, 1971 (1982).

D. J. Thouless, P. W. Anderson and R. G. Palmer, Solution of a 'Solvable Model of a Spin Glass', Phil. Mag. **35**, 593 (1977).

C. K. I. Williams, Computing with Infinite Networks, in *Neural Information Processing Systems 9*, M. C. Mozer, M. I. Jordan and T. Petsche, eds., 295-301. MIT Press (1997).

C. K. I. Williams and C. E. Rasmussen, Gaussian Processes for Regression, in *Neural Information Processing Systems 8*, D. S. Touretzky, M. C. Mozer and M. E. Hasselmo eds., 514-520, MIT Press (1996).

K. Y. M. Wong, Microscopic Equations and Stability Conditions in Optimal Neural Networks, Europhys. Lett. **30**, 245 (1995).
